# A Generalized Bradley-Terry Model: From Group Competition to Individual Skill

**Tzu-Kuo Huang**     **Chih-Jen Lin**
Department of Computer Science
National Taiwan University
Taipei 106, Taiwan

**Ruby C. Weng**
Department of Statistics
National Chenechi University
Taipei 116, Taiwan

## Abstract

The Bradley-Terry model for paired comparison has been popular in many areas. We propose a generalized version in which paired individual comparisons are extended to paired team comparisons. We introduce a simple algorithm with convergence proofs to solve the model and obtain individual skill. A useful application to multi-class probability estimates using error-correcting codes is demonstrated.

## 1 Introduction

The Bradley-Terry model [2] for paired comparisons has been broadly applied in many areas such as statistics, sports, and machine learning. It considers the model

$$P(\text{individual } i \text{ beats individual } j) = \frac{\pi_i}{\pi_i + \pi_j}, \tag{1}$$

where $\pi_i$ is the overall skill of the $i$th individual. Given $k$ individuals and $r_{ij}$ as the number of times that $i$ beats $j$, an approximate skill $p_i$ can be found by minimizing the negative log likelihood of the model (1):

$$\min_{\mathbf{p}} \quad l(\mathbf{p}) = -\sum_{i<j} \left( r_{ij} \log \frac{p_i}{p_i + p_j} + r_{ji} \log \frac{p_j}{p_i + p_j} \right)$$

$$\text{subject to} \quad 0 \leq p_i, i = 1, \ldots, k, \sum_{i=1}^{k} p_i = 1. \tag{2}$$

Thus, from paired comparisons, we can obtain individual performance. This model dates back to [14] and has been extended to more general settings. Some reviews are, for example, [5, 6]. Problem (2) can be solved by a simple iterative procedure:

**Algorithm 1**
1. Start with any initial $p_j^0 > 0$, $j = 1, \ldots, k$.
2. Repeat $(t = 0, 1, \ldots)$
   a. Let $s = (t \mod k) + 1$. For $j = 1, \ldots, k$, define

$$p_j^{t,n} \equiv \begin{cases} \dfrac{\sum_{i:i\neq s} r_{si}}{\sum_{i:i\neq s} \frac{r_{si}+r_{is}}{p_s^t + p_i^t}} & \text{if } j = s, \\ p_j^t & \text{if } j \neq s. \end{cases} \tag{3}$$

      b. Normalize $\mathbf{p}^{t,n}$ to be $\mathbf{p}^{t+1}$.
   until $\partial l(\mathbf{p}^t)/\partial p_j = 0, j = 1, \ldots, k$ are satisfied.

This algorithm is so simple that there is no need to use sophisticated optimization techniques. If $r_{ij} > 0, \forall i, j$, Algorithm 1 globally converges to the unique minimum of (2). A systematic study of the convergence is in [9].

Several machine learning work have used the Bradley-Terry model and one is to obtain multi-class probability estimates from pairwise coupling [8]. For any data instance $\mathbf{x}$, if $n_{ij}$ is the number of training data in the $i$th or $j$th class, and

$$r_{ij} \approx n_{ij} P(\mathbf{x} \text{ in class } i \mid \mathbf{x} \text{ in class } i \text{ or } j)$$

is available, solving (2) obtains the estimate of $P(\mathbf{x} \text{ in class } i), i = 1, \ldots, k$. [13] tried to extend this algorithm to other multi-class settings such as "one-against-the rest" or "error-correcting coding," but did not provide a convergence proof. In Section 5.2 we show that the algorithm proposed in [13] indeed has some convergence problems.

In this paper, we propose a generalized Bradley-Terry model where each comparison is between two disjoint subsets of subjects. Then from the results of team competitions, we can approximate the skill of each individual. This model has many potential applications. For example, from records of tennis or badminton doubles (or singles and doubles combined), we may obtain the rank of all individuals. A useful application in machine learning is multi-class probability estimates using error-correcting codes. We then introduce a simple iterative method to solve the generalized model with a convergence proof. Experiments on multi-class probability estimates demonstrate the viability of the proposed model and algorithm. Due to space limitation, we omit all proofs in this paper.

## 2 Generalized Bradley-Terry Model

We propose a generalized Bradley-Terry model where, using team competition results, we can approximate individual skill levels. Consider a group of $k$ individuals: $\{1, \ldots, k\}$. Two disjoint subsets $I_i^+$ and $I_i^-$ form teams for games and $r_i \geq 0$ ($r_i' \geq 0$) is the number of times that $I_i^+$ beats $I_i^-$ ($I_i^-$ beats $I_i^+$). Thus, we have $I_i \subset \{1, \ldots, k\}, i = 1, \ldots, m$ so that

$$I_i = I_i^+ \cup I_i^-, \qquad I_i^+ \neq \emptyset, I_i^- \neq \emptyset, \text{ and } I_i^+ \cap I_i^- = \emptyset.$$

Under the model that

$$P(I_i^+ \text{ beats } I_i^-) = \frac{\sum_{j \in I_i^+} \pi_j}{\sum_{j \in I_i^+} \pi_j + \sum_{j \in I_i^-} \pi_j} = \frac{\sum_{j \in I_i^+} \pi_j}{\sum_{j \in I_i} \pi_j},$$

we can define

$$q_i \equiv \sum_{j \in I_i} p_j, \qquad q_i^+ \equiv \sum_{j \in I_i^+} p_j, \qquad q_i^- \equiv \sum_{j \in I_i^-} p_j,$$

and minimize the negative log likelihood

$$\min_{\mathbf{p}} \quad l(\mathbf{p}) = -\sum_{i=1}^{m} \left( r_i \log(q_i^+/q_i) + r_i' \log(q_i^-/q_i) \right), \tag{4}$$

under the same constraints of (2). If $I_i, i = 1, \ldots, k(k-1)/2$ are as the following:

| $I_i^+$ | $I_i^-$ | $r_i$ | $r_i'$ |
|---|---|---|---|
| $\{1\}$ | $\{2\}$ | $r_{12}$ | $r_{21}$ |
| $\vdots$ | $\vdots$ | $\vdots$ | $\vdots$ |
| $\{k-1\}$ | $\{k\}$ | $r_{k-1,k}$ | $r_{k,k-1}$ |

then (4) goes back to (2). The difficulty of solving (4) over solving (2) is that now $l(\mathbf{p})$ is expressed in terms of $q_i^+, q_i^-, q_i$ but the real variable is $\mathbf{p}$. The original Bradley-Terry model is a special case of other statistical models such as log-linear or generalized linear model, so methods other than Algorithm 1 (e.g., iterative scaling and iterative weighted least squares) can also be used. However, (4) is not in a form of such models and hence these methods cannot be applied. We propose the following algorithm to solve (4).

**Algorithm 2**
  1. Start with $p_j^0 > 0, j = 1, \ldots, k$ and corresponding $q_i^{0,+}, q_i^{0,-}, q_i^0, i = 1, \ldots, m$.
  2. Repeat $(t = 0, 1, \ldots)$
     a. Let $s = (t \mod k) + 1$. For $j = 1, \ldots, k$, define

$$p_j^{t,n} \equiv \begin{cases} \dfrac{\sum_{i:s\in I_i^+} \frac{r_i}{q_i^{t,+}} + \sum_{i:s\in I_i^-} \frac{r_i'}{q_i^{t,-}}}{\sum_{i:s\in I_i} \frac{r_i+r_i'}{q_i^t}} p_j^t & \text{if } j = s, \\ p_j^t & \text{if } j \neq s. \end{cases} \qquad (5)$$

     b. Normalize $\mathbf{p}^{t,n}$ to $\mathbf{p}^{t+1}$.
     c. Update $q_i^{t,+}, q_i^{t,-}, q_i^t$ to $q_i^{t+1,+}, q_i^{t+1,-}, q_i^{t+1}, i = 1, \ldots, m$.
     until $\partial l(\mathbf{p}^t)/\partial p_j = 0, j = 1, \ldots, k$ are satisfied.

For the multiplicative factor in (5) to be well defined (i.e., non-zero denominator), we need Assumption 1, which will be discussed in Section 3. Eq. (5) is a simple fixed-point type update; in each iteration, only one component (i.e., $p_s^t$) is modified while the others remain the same. It is motivated from using a descent direction to strictly decrease $l(\mathbf{p})$: If $\partial l(\mathbf{p}^t)/\partial p_s \neq 0$, then

$$\frac{\partial l(\mathbf{p}^t)}{\partial p_s} \cdot (p_s^{t,n} - p_s^t) = \left( -\left( \frac{\partial l(\mathbf{p}^t)}{\partial p_s} \right)^2 p_s^t \right) \Big/ \left( \sum_{i:s\in I_i} \frac{r_i + r_i'}{q_i^t} \right) < 0, \qquad (6)$$

where

$$\frac{\partial l(\mathbf{p})}{\partial p_s} = -\sum_{i:s\in I_i^+} \frac{r_i}{q_i^+} - \sum_{i:s\in I_i^-} \frac{r_i'}{q_i^-} + \sum_{i:s\in I_i} \frac{r_i + r_i'}{q_i}.$$

Thus, $p_s^{t,n} - p_s^t$ is a descent direction in optimization since a sufficiently small step along this direction guarantees the strict decrease of the function value. Since now we take the whole direction without searching for the step size, more efforts are needed to prove the strict decrease in Lemma 1. However, (6) does hint that (5) is a reasonable update.

**Lemma 1** *If $p_s^t > 0$ is the index to be updated and $\partial l(\mathbf{p}^t)/\partial p_s \neq 0$, then $l(\mathbf{p}^{t+1}) < l(\mathbf{p}^t)$.*

If we apply the update rule (5) on the pairwise model,

$$\frac{\sum_{i:i\neq s} \frac{r_{si}}{p_s^t}}{\sum_{i:i\neq s} \frac{r_{si}}{p_s^t+p_i^t} + \sum_{i:i\neq s} \frac{r_{is}}{p_s^t+p_i^t}} p_s^t = \frac{\sum_{i:i\neq s} r_{si}}{\sum_{i:i\neq s} \frac{r_{si}+r_{is}}{p_s^t+p_i^t}} \text{ and (5) goes back to (3).}$$

## 3  Convergence of Algorithm 2

For any point satisfying $\partial l(\mathbf{p})/\partial p_j = 0, j = 1, \ldots, k$ and constraints of (4), it is a stationary point of (4)[1]. We will prove that Algorithm 2 converges to such a point. If

it stops in a finite number of iterations, then $\partial l(\mathbf{p})/\partial p_j = 0, j = 1, \ldots, k$, which means a stationary point of (4) is already obtained. Thus, we only need to handle the case where $\{\mathbf{p}^t\}$ is an infinite sequence. As $\{\mathbf{p}^t\}_{t=0}^{\infty}$ is in a compact (i.e., closed and bounded) set $\{\mathbf{p} \mid 0 \leq p_j \leq 1, \sum_{j=1}^{k} p_j = 1\}$, it has at least one convergent subsequence. Assume $\mathbf{p}^*$ is one such convergent point. In the following we will prove that $\partial l(\mathbf{p}^*)/\partial p_j = 0, j = 1, \ldots, k$.

To prove the convergence of a fixed-point type algorithm, we need that if $p_s^* > 0$ and $\partial l(\mathbf{p}^*)/\partial p_s \neq 0$, then from $p_s^*$ we can use (5) to update it to $p_s^{*,n} \neq p_s^*$. We thus make the following assumption to ensure that $\mathbf{p}_s^* > 0$ (see also Theorem 1).

**Assumption 1** *For each $j \in \{1, \ldots, k\}$,*

$\cup_{i:i \in A} I_i = \{1, \ldots, k\}$, *where $A = \{i \mid (I_i^+ = \{j\}, r_i > 0) \text{ or } (I_i^- = \{j\}, r_i' > 0)\}$.*

*That is, each individual forms a winning (losing) team in some competitions which together involve all subjects.*

An issue left in Section 2 is whether the multiplicative factor in (5) is well defined. With Assumption 1 and initial $p_j^0 > 0, j = 1, \ldots, k$, one can show by induction that $p_j^t > 0, \forall t$ and hence the denominator of (5) is never zero: If $p_j^t > 0$, Assumption 1 implies that $\sum_{i:j \in I_i^+} r_i/q_i^{t,+}$ or $\sum_{i:j \in I_i^-} r_i'/q_i^{t,-}$ is positive. Thus, both numerator and denominator in the multiplicative factor are positive, and so is $p_j^{t+1}$.

If $r_{ij} > 0$, the original Bradley-Terry model satisfies Assumption 1. No matter the model satisfies the assumption or not, an easy way to fulfill it is to add an additional term

$$-\mu \sum_{s=1}^{k} \log \left( \frac{p_s}{\sum_{j=1}^{k} p_j} \right) \tag{7}$$

to $l(\mathbf{p})$, where $\mu$ is a small positive number. That is, for each $s$, we make an $I_i = \{1, \ldots, k\}$ with $I_i^+ = \{s\}, r_i = \mu$, and $r_i' = 0$. As $\sum_{j=1}^{k} p_j = 1$ is one of the constraints, (7) reduces to $-\mu \sum_{s=1}^{k} \log p_s$, which is a barrier term in optimization to ensure that $p_s$ does not go to zero. The property $p_s^* > 0$ and the convergence of Algorithm 2 are in Theorem 1:

**Theorem 1** *Under Assumption 1, any convergent point $\mathbf{p}^*$ of Algorithm 2 satisfies $p_s^* > 0, s = 1, \ldots, k$ and is a stationary point of (4).*

## 4 Asymptotic Distribution of the Maximum Likelihood Estimator

For the standard Bradley-Terry model, asymptotic distribution of the MLE (i.e., $\mathbf{p}$) has been discussed in [5]. In this section, we discuss the asymptotic distribution for the proposed estimator. To work on the real probability $\boldsymbol{\pi}$, we define

$$\bar{q}_i \equiv \sum_{j \in I_i} \pi_j, \qquad \bar{q}_i^+ \equiv \sum_{j \in I_i^+} \pi_j, \qquad \bar{q}_i^- \equiv \sum_{j \in I_i^-} \pi_j,$$

and consider $n_i \equiv r_i + r_i'$ as a constant. Note that $r_i \sim \text{BIN}(n_i, \bar{q}_i^+/\bar{q}_i)$ is a random variable representing the number of times that $I_i^+$ beats $I_i^-$ in $n_i$ competitions. By defining for $s, t = 1, \ldots, k$,

$$\lambda_{ss} \equiv \text{var} \left[ \frac{\partial l(\boldsymbol{\pi})}{\partial p_s} \right] = \sum_{i:s \in I_i^+} \frac{n_i \bar{q}_i^-}{\bar{q}_i^+ \bar{q}_i^2} + \sum_{i:s \in I_i^-} \frac{n_i \bar{q}_i^+}{\bar{q}_i^- \bar{q}_i^2},$$

$$\lambda_{st} \equiv \text{cov} \left[ \frac{\partial l(\boldsymbol{\pi})}{\partial p_s}, \frac{\partial l(\boldsymbol{\pi})}{\partial p_t} \right] = \sum_{i:s,t \in I_i^+} \frac{\bar{q}_i^- n_i}{\bar{q}_i^+ \bar{q}_i^2} -$$

$$\sum_{i:(s,t) \in I_i^+ \times I_i^-} \frac{n_i}{\bar{q}_i^2} - \sum_{i:(s,t) \in I_i^- \times I_i^+} \frac{n_i}{\bar{q}_i^2} + \sum_{i:s,t \in I_i^-} \frac{\bar{q}_i^+ n_i}{\bar{q}_i^- \bar{q}_i^2}, s \neq t,$$

we have the following theorem:

**Theorem 2** *Let $n$ be the total number of comparisons. If $r_i$ is independent of $r_j, \forall i \neq j$, then $\sqrt{n}(p_1 - \pi_1), \ldots, \sqrt{n}(p_{k-1} - \pi_{k-1})$ have for large samples the multivariate normal distribution with zero means and dispersion matrix $[\lambda'_{st}]^{-1}$, where*

$$\lambda'_{st} = \lambda_{st} - \lambda_{sk} - \lambda_{tk} + \lambda_{kk}, s, t = 1, \ldots, k-1.$$

## 5 Application to Multi-class Probability Estimates

Many classification methods are two-class based approaches and there are different ways to extend them for multi-class cases. Most existing studies focus on predicting class labels but not probability estimates. In this section, we discuss how the generalized Bradley-Terry model can be applied to multi-class probability estimates.

Error-correction coding [7, 1] is a general method to construct binary classifiers and combine them for multi-class prediction. It suggests some ways to construct $I_i^+$ and $I_i^-$; both are subsets of $\{1, \ldots, k\}$. Then one trains a binary model using data from classes in $I_i^+$ ($I_i^-$) as positive (negative). Simple and commonly used methods such as "one-against-one" and "one-against-the rest" are its special cases. Given $n_i$ the number of training data with classes in $I_i = I_i^+ \cup I_i^-$, we assume here that for any data $\mathbf{x}$,

$$r_i \approx n_i P(\mathbf{x} \text{ in classes of } I_i^+ \mid \mathbf{x} \text{ in classes of } I_i^+ \text{ or } I_i^-) \tag{8}$$

is available, and the task is to approximate $P(\mathbf{x} \text{ in class } s), s = 1, \ldots, k$. In the rest of this section we discuss the special case "one-against-the rest" and the earlier results in [13].

### 5.1 Properties of the "One-against-the rest" Approach

For this approach, $I_i, i = 1, \ldots, m$ are

| $I_i^+$ | $I_i^-$ | $r_i$ | $r_i'$ |
|---------|---------|-------|--------|
| $\{1\}$ | $\{2, \ldots, k\}$ | $r_1$ | $1 - r_1$ |
| $\{2\}$ | $\{1, 3, \ldots, k\}$ | $r_2$ | $1 - r_2$ |
| $\vdots$ | $\vdots$ | $\vdots$ | $\vdots$ |

Now $n_1 = \cdots = n_m =$ the total number of training data, so the solution to (4) is not affected by $n_i$. Thus, we remove it from (8), so $r_i + r_i' = 1$. As $\partial l(\mathbf{p})/\partial p_s = 0$ becomes

$$\frac{r_s}{p_s} + \sum_{j:j\neq s} \frac{r_j'}{1 - p_j} = k, \text{ we have } \frac{r_1}{p_1} - \frac{1 - r_1}{1 - p_1} = \cdots = \frac{r_k}{p_k} - \frac{1 - r_k}{1 - p_k} = k - \sum_{j=1}^{k} \frac{r_j'}{1 - p_j} = \delta,$$

where $\delta$ is a constant. These equalities provide another way to solve $\mathbf{p}$, and $p_s = ((1 + \delta) - \sqrt{(1+\delta)^2 - 4r_s\delta})/2\delta$. Note that $((1 + \delta) + \sqrt{(1+\delta)^2 - 4r_s\delta})/2\delta$ also satisfies the equalities, but it is negative when $\delta < 0$, and greater than 1 when $\delta > 0$. By solving $\sum_{s=1}^{m} p_s = 1$, we obtain $\delta$ and the optimal $\mathbf{p}$.

From the formula of $p_s$, if $\delta > 0$, larger $p_s$ implies smaller $(1+\delta)^2 - 4r_s\delta$ and hence larger $r_s$. It is similar for $\delta < 0$. Thus, the order of $p_1, \ldots, p_k$ is the same as that of $r_1, \ldots, r_k$:

**Theorem 3** *If $r_s \geq r_t$, then $p_s \geq p_t$.*

### 5.2 The Approach in [13] for Error-Correcting Codes

[13] was the first attempt to address the probability estimates using general error-correcting codes. By considering the same optimization problem (4), it proposes a heuristic update rule

$$p_s^{t,n} \equiv \frac{\sum_{i:s\in I_i^+} r_i + \sum_{i:s\in I_i^-} r_i'}{\sum_{i:s\in I_i^+} \frac{n_i q_i^{t,+}}{q_i^t} + \sum_{i:s\in I_i^-} \frac{n_i q_i^{t,-}}{q_i^t}} p_s^t, \tag{9}$$

but does not provide a convergence proof. For a fixed-point update, we expect that at the optimum, the multiplicative factor in (9) is one. However, unlike (5), when the factor is one, (9) does not relate to $\partial l(\mathbf{p})/\partial p_s = 0$. In fact, a simple example shows that this algorithm may never converge. Taking the "one-against-the rest" approach, if we keep $\sum_{i=1}^{k} p_i^t = 1$ and assume $n_i = 1$, then $r_i + r_i' = 1$ and the factor in the update rule (9) is

$$\frac{r_s + \sum_{i:i\neq s} r_i'}{p_s^t + \sum_{i:i\neq s}(1-p_i^t)} = \frac{k-1+2r_s - \sum_{i=1}^{k} r_i}{k-2+2p_s^t}.$$

If the algorithm converges and the factor approaches one, then $p_s = (1 + 2r_s - \sum_{i=1}^{k} r_i)/2$ but they may not satisfy $\sum_{s=1}^{k} p_s = 1$. Therefore, if in the algorithm we keep $\sum_{i=1}^{k} p_i^t = 1$ as [13] did, the factor may not approach one and the algorithm does not converge. More generally, if $I_i = \{1, \ldots, k\}, \forall i$, the algorithm may not converge. As $q_i^t = 1$, the condition that the factor equals one can be written as a linear equation of $\mathbf{p}$. Together with $\sum_{i=1}^{k} p_i = 1$, there is an over-determined linear system (i.e., $k+1$ equations and $k$ variables).

## 6 Experiments on Multi-class Probability Estimates

### 6.1 Simulated Examples

We consider the same settings in [8, 12] by defining three possible class probabilities:
(a) $p_1 = 1.5/k$, $p_j = (1 - p_1)/(k - 1)$, $j = 2, \ldots, k$.
(b) $k_1 = k/2$ if $k$ is even, and $(k + 1)/2$ if $k$ is odd; then $p_1 = 0.95 \times 1.5/k_1$, $p_i = (0.95 - p_1)/(k_1 - 1)$ for $i = 2, \ldots, k_1$, and $p_i = 0.05/(k - k_1)$ for $i = k_1 + 1, \ldots, k$.
(c) $p_1 = 0.95 \times 1.5/2$, $p_2 = 0.95 - p_1$, and $p_i = 0.05/(k - 2)$, $i = 3, \ldots, k$.
Classes are competitive in case (a), but only two dominate in case (c). We then generate $r_i$ by adding some noise to $q_i^+/q_i$:

$$r_i = \min(\max(\epsilon, \frac{q_i^+}{q_i}(1 + 0.1N(0, 1))), 1 - \epsilon).$$

Then $r_i' = 1 - r_i$. Here $\epsilon = 10^{-7}$ is used so that all $r_i, r_i'$ are positive. We consider the four encodings used in [1] to generate $I_i$:
1. "1vs1": the pairwise approach (eq. (2)).
2. "1vsrest": the "one-against-the rest" approach in Section 5.1.
3. "dense": $I_i = \{1, \ldots, k\}$ for all $i$. $I_i$ is randomly split to two equally-sized sets $I_i^+$ and $I_i^-$. $[10 \log_2 k]$ such splits are generated[2]. Following [1], we repeat this procedure 100 times and select the one whose $[10 \log_2 k]$ splits have the smallest distance.
4. "sparse": $I_i^+, I_i^-$ are randomly drawn from $\{1, \ldots, k\}$ with $E(|I_i^+|) = E(|I_i^-|) = k/4$. Then $[15 \log_2 k]$ such splits are generated. Similar to "dense," we repeat the procedure 100 times to find a good coding.

Figure 1 shows averaged accuracy rates over 500 replicates for each of the four methods when $k = 2^2, 2^3, \ldots, 2^6$. "1vs1" is good for (a) and (b), but suffers some losses in (c), where the class probabilities are highly unbalanced. [12] has observed this and proposed some remedies. "1vsrest" is quite competitive in all three scenarios. Furthermore, "dense" and "sparse" are less competitive in cases (a) and (b) when $k$ is large. Due to the large $|I_i^+|$ and $|I_i^-|$, the model is unable to single out a clear winner when probabilities are more balanced. We also analyze the (relative) mean square error (MSE) in Figure 2:

$$\text{MSE} = \frac{1}{500} \sum_{j=1}^{500} \left( \sum_{i=1}^{k} (\hat{p}_i^j - p_i)^2 / \sum_{i=1}^{k} p_i^2 \right), \tag{10}$$

where $\hat{\mathbf{p}}^j$ is the probability estimate obtained in the $j$th of the 500 replicates. Results of Figures 2(b) and 2(c) are consistent with those of the accuracy. Note that in Figure 2(a), as $\mathbf{p}$ (and $\hat{\mathbf{p}}^j$) are balanced, $\sum_{i=1}^{k} (\hat{p}_i^j - p_i)^2$ is small. Hence, all approaches have small MSE though some have poor accuracy.

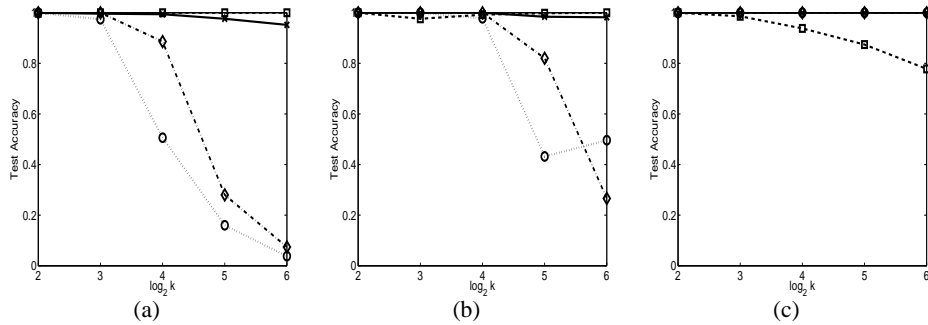

Figure 1: Accuracy by the four encodings: "1vs1" (dashed line, square), "1vsrest" (solid line, cross), "dense" (dotted line, circle), "sparse" (dashdot line, asterisk)

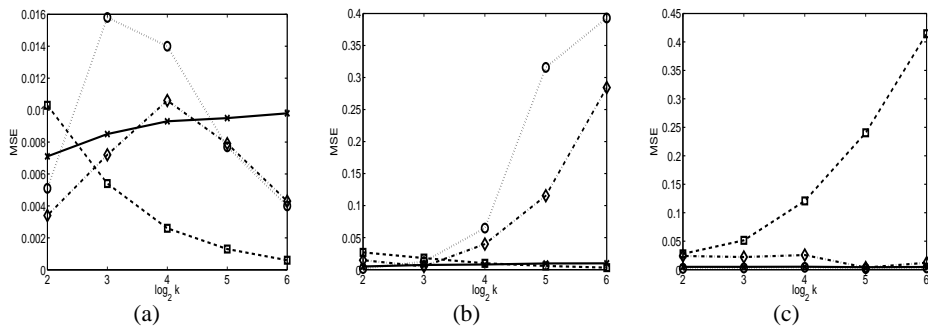

Figure 2: MSE by the four encodings: legend the same as Figure 1

## 6.2 Experiments on Real Data

In this section we present experimental results on some real-world multi-class problems. They have been used in [12], which provides more information about data preparation. Two problem sizes, 300/500 and 800/1,000 for training/testing, are used. 20 training/testing splits are generated and the testing error rates are averaged. All data used are available at http://www.csie.ntu.edu.tw/~cjlin/papers/svmprob/data. We use the same four ways in Section 6.1 to generate $I_i$. All of them have $|I_1| \approx \cdots \approx |I_m|$. With the property that these multi-class problems are reasonably balanced, we set $n_i = 1$ in (8).

Since there are no probability values available for these problems, we compare the accuracy by predicting the label with the largest probability estimate. The purpose here is to compare the four probability estimates but not to check the difference from existing multi-class classification techniques. We consider support vector machines (SVM) [4] with the RBF kernel as the binary classifier. An improved version [10] of [11] obtains $r_i$. Full SVM parameter selection is conducted before testing, although due to space limitation, we omit details here. The code is modified from LIBSVM [3], a library for support vector machines. The resulting accuracy is in Table 1 for smaller and larger training/testing sets. Except "1vs1," the other three approaches are quite competitive. These results indicate that practical problems are more similar to the case of (c) in Section 6.1, where few classes dominate. This observation is consistent with the findings in [12]. Moreover, "1vs1" suffers some losses when $k$ is larger (e.g., letter), the same as in Figure 1(c); so for "1vs1," [12] proposed using a quadratic model instead of the Bradley-Terry model.

In terms of the computational time, because the number of binary problems for "dense" and "sparse" ($[10 \log_2 k]$ and $[15 \log_2 k]$, respectively) is larger than $k$, and each binary problem

involves many classes of data (all and one half), their training time is longer than "1vs1" and "1vsrest." "Dense" is particularly time consuming. Note that though "1vs1" solves $k(k-1)/2$ binaries, it is efficient as each binary problem involves only two classes of data.

Table 1: Average of 20 test errors (in percentage) by four encodings (lowest boldfaced)

| Problem | $k$ | 300 training and 500 testing | | | | 800 training and 1,000 testing | | | |
|---|---|---|---|---|---|---|---|---|---|
| | | 1vs1 | 1vsrest | dense | sparse | 1vs1 | 1vsrest | dense | sparse |
| dna | 3 | 10.47 | 10.33 | 10.45 | **10.19** | **6.21** | 6.45 | 6.415 | 6.345 |
| waveform | 3 | **15.01** | 15.35 | 15.66 | 15.12 | **13.525** | 13.635 | 13.76 | 13.99 |
| satimage | 6 | **14.22** | 15.08 | 14.72 | 14.8 | **11.54** | 11.74 | 11.865 | 11.575 |
| segment | 7 | 6.24 | 6.69 | 6.62 | **6.19** | 3.295 | 3.605 | 3.52 | **3.25** |
| USPS | 10 | 11.37 | 10.89 | **10.81** | 11.14 | 7.78 | 7.49 | **7.31** | 7.575 |
| MNIST | 10 | 13.84 | 12.56 | 13.0 | **12.29** | 8.11 | **7.37** | 7.59 | 7.535 |
| letter | 26 | 39.73 | 35.17 | **33.86** | 33.88 | 21.11 | 19.685 | 20.14 | **19.49** |

In summary, we propose a generalized Bradley-Terry model which gives individual skill from group competition results. A useful application to general multi-class probability estimate is demonstrated.

## Footnotes

[1]A stationary point means a Karash-Kunh-Tucker (KKT) point for constrained optimization problems like (2) and (4). Note that here $\partial l(\mathbf{p})/\partial p_j = 0$ implies (and is more restricted than) the KKT condition.

[2]We use $[x]$ to denote the nearest integer value of $x$.

# References

[1] E. L. Allwein, R. E. Schapire, and Y. Singer. Reducing multiclass to binary: a unifying approach for margin classifiers. *Journal of Machine Learning Research*, 1:113–141, 2001.

[2] R. A. Bradley and M. Terry. The rank analysis of incomplete block designs: I. the method of paired comparisons. *Biometrika*, 39:324–345, 1952.

[3] C.-C. Chang and C.-J. Lin. *LIBSVM: a library for support vector machines*, 2001. Software available at `http://www.csie.ntu.edu.tw/~cjlin/libsvm`.

[4] C. Cortes and V. Vapnik. Support-vector network. *Machine Learning*, 20:273–297, 1995.

[5] H. A. David. *The method of paired comparisons*. Oxford University Press, New York, second edition, 1988.

[6] R. R. Davidson and P. H. Farquhar. A bibliography on the method of paired comparisons. *Biometrics*, 32:241–252, 1976.

[7] T. G. Dietterich and G. Bakiri. Solving multiclass learning problems via error-correcting output codes. *Journal of Artificial Intelligence Research*, 2:263–286, 1995.

[8] T. Hastie and R. Tibshirani. Classification by pairwise coupling. In M. I. Jordan, M. J. Kearns, and S. A. Solla, editors, *Advances in Neural Information Processing Systems 10*. MIT Press, Cambridge, MA, 1998.

[9] D. R. Hunter. MM algorithms for generalized Bradley-Terry models. *The Annals of Statistics*, 32:386–408, 2004.

[10] H.-T. Lin, C.-J. Lin, and R. C. Weng. A note on Platt's probabilistic outputs for support vector machines. Technical report, Department of Computer Science, National Taiwan University, 2003.

[11] J. Platt. Probabilistic outputs for support vector machines and comparison to regularized likelihood methods. In A. Smola, P. Bartlett, B. Schölkopf, and D. Schuurmans, editors, *Advances in Large Margin Classifiers*, Cambridge, MA, 2000. MIT Press.

[12] T.-F. Wu, C.-J. Lin, and R. C. Weng. Probability estimates for multi-class classification by pairwise coupling. In S. Thrun, L. Saul, and B. Schölkopf, editors, *Advances in Neural Information Processing Systems 16*. MIT Press, Cambridge, MA, 2004.

[13] B. Zadrozny. Reducing multiclass to binary by coupling probability estimates. In T. G. Dietterich, S. Becker, and Z. Ghahramani, editors, *Advances in Neural Information Processing Systems 14*, pages 1041–1048. MIT Press, Cambridge, MA, 2002.

[14] E. Zermelo. Die berechnung der turnier-ergebnisse als ein maximumproblem der wahrscheinlichkeitsrechnung. *Mathematische Zeitschrift*, 29:436–460, 1929.
